# Multivariate Dyadic Regression Trees for Sparse Learning Problems

**Han Liu** and **Xi Chen**
School of Computer Science, Carnegie Mellon University
Pittsburgh, PA 15213

## Abstract

We propose a new nonparametric learning method based on multivariate dyadic regression trees (MDRTs). Unlike traditional dyadic decision trees (DDTs) or classification and regression trees (CARTs), MDRTs are constructed using penalized empirical risk minimization with a novel sparsity-inducing penalty. Theoretically, we show that MDRTs can simultaneously adapt to the unknown sparsity and smoothness of the true regression functions, and achieve the nearly optimal rates of convergence (in a minimax sense) for the class of $(\alpha, C)$-smooth functions. Empirically, MDRTs can simultaneously conduct function estimation and variable selection in high dimensions. To make MDRTs applicable for large-scale learning problems, we propose a greedy heuristics. The superior performance of MDRTs are demonstrated on both synthetic and real datasets.

## 1 Introduction

Many application problems need to simultaneously predict several quantities using a common set of variables, e.g. predicting multi-channel signals within a time frame, predicting concentrations of several chemical constitutes using the mass spectra of a sample, or predicting expression levels of many genes using a common set of phenotype variables. These problems can be naturally formulated in terms of multivariate regression.

In particular, let $\left\{ (\mathbf{x}^1, \mathbf{y}^1), \ldots, (\mathbf{x}^n, \mathbf{y}^n) \right\}$ be $n$ independent and identically distributed pairs of data with $\mathbf{x}^i \in \mathcal{X} \subset \mathbb{R}^d$ and $\mathbf{y}^i \in \mathcal{Y} \subset \mathbb{R}^p$ for $i = 1, \ldots, n$. Moreover, we denote the $j$th dimension of $\mathbf{y}$ by $\mathbf{y}_j = (\mathbf{y}_j^1, \ldots, \mathbf{y}_j^n)^T$ and $k$th dimension of $\mathbf{x}$ by $\mathbf{x}_k = (\mathbf{x}_k^1, \ldots, \mathbf{x}_k^n)^T$. Without loss of generality, we assume $\mathcal{X} = [0, 1]^d$ and the true model on $\mathbf{y}_j$ is :

$$\mathbf{y}_j^i = f_j(\mathbf{x}^i) + \epsilon_j^i, \ i = 1, \ldots, n, \tag{1}$$

where $f_j : \mathbb{R}^d \to \mathbb{R}$ is a smooth function. In the sequel, let $f = (f_1, \ldots, f_p)$, where $f : \mathbb{R}^d \to \mathbb{R}^p$ is a $p$-valued smooth function. The vector form of (1) then becomes $\mathbf{y}^i = f(\mathbf{x}^i) + \boldsymbol{\epsilon}^i, \ i = 1, \ldots, n$. We also assume that the noise terms $\left\{ \epsilon_j^i \right\}_{i,j}$ are independently distributed and bounded almost surely.

This is a general setting of the nonparametric multivariate regression. From the minimax theory, we know that estimating $f$ in high dimensions is very challenging. For example, when $f_1, \ldots, f_p$ lie in a $d$-dimensional Sobolev ball with order $\alpha$ and radius $C$, the best convergence rate for the minimax risk is $p \cdot n^{-2\alpha/(2\alpha+d)}$. For a fixed $\alpha$, such rate can be very slow when $d$ becomes large.

However, in many real world applications, the true regression function $f$ may depend only on a small set of variables. In other words, the problem is *jointly sparse*:

$$f(\mathbf{x}) = f(\mathbf{x}_S) = (f_1(\mathbf{x}_S), \ldots, f_p(\mathbf{x}_S)),$$

where $\mathbf{x}_S = (\mathbf{x}_k : k \in S), S \subset \{1, \ldots, d\}$ is a subset of covariates with size $r = |S| \ll d$. If $S$ has been given, the minimax lower bound can be improved to be $p \cdot n^{-2\alpha/(2\alpha+r)}$, which is the best possible rate can be expected. For sparse learning problems, our task is to develop an estimator, which adaptively achieves this faster rate of convergence without knowing $S$ in advance.

Previous research on these problems can be roughly divided into three categories: (i) parametric linear models, (ii) nonparametric additive models, and (iii) nonparametric tree models. The methods in the first category assume that the true models are linear and use some block-norm regularization to induce jointly sparse solutions [16, 11, 13, 5]. If the linear model assumptions are correct, accurate estimates can be obtained. However, given the increasing complexity of modern applications, conclusions inferred under these restrictive linear model assumptions can be misleading. Recently, significant progress has been made on inferring nonparametric additive models with joint sparsity constraints [7, 10]. For additive models, each $f_j(\mathbf{x})$ is assumed to have an additive form: $f_j(\mathbf{x}) = \sum_{k=1}^{d} f_{jk}(\mathbf{x}_k)$. Although they are more flexible than linear models, the additivity assumptions might still be too stringent for real world applications.

A family of more flexible nonparametric methods are based on tree models. One of the most popular tree methods is the classification and regression tree (CART) [2]. It first grows a full tree by orthogonally splitting the axes at locally optimal splitting points, then prunes back the full tree to form a subtree. Theoretically, CART is hard to analyze unless strong assumptions have been enforced [8]. In contrast to CART, dyadic decision trees (DDTs) are restricted to only axis-orthogonal dyadic splits, i.e. each dimension can only be split at its midpoint. For a broad range of classification problems, [15] showed that DDTs using a special penalty can attain nearly optimal rate of convergence in a minimax sense. [1] proposed a dynamic programming algorithm for constructing DDTs when the penalty term has an additive form, i.e. the penalty of the tree can be written as the sum of penalties on all terminal nodes. Though intensively studied for classification problems, the dyadic decision tree idea has not drawn much attention in the regression settings. One of the closest results we are aware of is [4], in which a single response dyadic regression procedure is considered for non-sparse learning problems. Another interesting tree model, "Bayesian Additive Regression Trees (BART)", is proposed under Bayesian framework [6], which is essentially a "sum-of-trees" model. Most of the existing work adopt the number of terminal nodes as the penalty. Such penalty cannot lead to sparse models since a tree with a small number of terminal nodes might still involve too many variables.

To obtain sparse models, we propose a new nonparametric method based on multivariate dyadic regression trees (MDRTs). Similar to DDTs, MDRTs are constructed using penalized empirical risk minimization. The novelty of MDRT is to introduce a sparsity-inducing term in the penalty, which explicitly induces sparse solutions. Our contributions are two-fold: (i) Theoretically, we show that MDRTs can simultaneously adapt to the unknown sparsity and smoothness of the true regression functions, and achieve the nearly optimal rate of convergence for the class of $(\alpha, C)$-smooth functions. (ii) Empirically, to avoid computationally prohibitive exhaustive search in high dimensions, we propose a two-stage greedy algorithm and its randomized version that achieve good performance in both function estimation and variable selection. Note that our theory and algorithm can be straightforwardly adapted to univariate sparse regression problem, which is a special case of the multivariate one. To the best of our knowledge, this is the first time such a sparsity-inducing penalty is equipped to tree models for solving sparse regression problems.

The rest of this paper is organized as follows. Section 2 presents MDRTs in detail. Section 3 studies the statistical properties of MDRTs. Section 4 presents the algorithms which approximately compute the MDRT solutions. Section 5 reports empirical results of MDRTs and their comparison with CARTs. Conclusions are made in Section 6.

## 2 Multivariate Dyadic Regression Trees

We adopt the notations in [15]. A MDRT $T$ is a multivariate regression tree that recursively divides the input space $\mathcal{X}$ by means of axis-orthogonal dyadic splits. The nodes of $T$ are associated with hyperrectangles (cells) in $\mathcal{X} = [0,1]^d$. The root node corresponds to $\mathcal{X}$ itself. If a node is associated to the cell $B = \prod_{j=1}^{d}[a_j, b_j]$, after being dyadically split on the dimension $k$, the two children are associated to the subcells $B^{k,1}$ and $B^{k,2}$:

$$B^{k,1} = \left\{ \mathbf{x}^i \in B \,|\, \mathbf{x}_k^i \leq \frac{a_k + b_k}{2} \right\} \text{ and } B^{k,2} = B \setminus B^{k,1}.$$

The set of terminal nodes of a MDRT $T$ is denoted as $\mathrm{term}(T)$. Let $B_t$ be the cell in $\mathcal{X}$ induced by a terminal node $t$, the partition induced by $\mathrm{term}(T)$ can be denoted as $\pi(T) = \{B_t | t \in \mathrm{term}(T)\}$.

For each terminal node $t$, we can fit a multivariate $m$-th order polynomial regression on data points falling in $B_t$. Instead of using all covariates, such a polynomial regression is only fitted on a set of active variables, which is denoted as $\mathcal{A}(t)$. For each node $b \in T$ (not necessarily a terminal node), $\mathcal{A}(b)$ can be an arbitrary subset of $\{1, \ldots, d\}$ satisfying two rules:

1. If a node is dyadically split perpendicular to the axis $k$, $k$ must belong to the active sets of its two children.
2. For any node $b$, let $\mathrm{par}(b)$ be its parent node, then $\mathcal{A}(\mathrm{par}(b)) \subset \mathcal{A}(b)$.

For a MDRT $T$, we define $\mathcal{F}_T^m$ to be the class of $p$-valued measurable $m$-th order polynomials corresponding to $\pi(T)$. Furthermore, for a dyadic integer $N = 2^L$, let $\mathcal{T}_N$ be the collection of all MDRTs such that no terminal cell has a side length smaller than $2^{-L}$.

Given integers $M$ and $N$, let $\mathcal{F}^{M,N}$ be defined as
$$\mathcal{F}^{M,N} = \cup_{0 \leq m \leq M} \cup_{T \in \mathcal{T}_N} \mathcal{F}_T^m.$$
The final MDRT estimator with respect to $\mathcal{F}^{M,N}$, denoted as $\widehat{f}^{M,N}$, can then be defined as
$$\widehat{f}^{M,N} = \underset{f \in \mathcal{F}^{M,N}}{\arg\min} \frac{1}{n} \sum_{i=1}^{n} \|\mathbf{y}^i - f(\mathbf{x}^i)\|_2^2 + \mathrm{pen}(f). \tag{2}$$
To define in detail $\mathrm{pen}(f)$ for $f \in \mathcal{F}^{M,N}$, let $T$ and $m$ be the MDRT and the order of polynomials corresponding to $f$, $\mathrm{pen}(f)$ then takes the following form:
$$\mathrm{pen}(f) = \lambda \cdot \frac{p}{n} \big( \log n (r_T + 1)^m (N_T + 1)^{r_T} + |\pi(T)| \log d \big), \tag{3}$$
where $\lambda > 0$ is a regularization parameter, $r_T = |\cup_{t \in \mathrm{term}(T)} \mathcal{A}(t)|$ corresponds to the number of relevant dimensions and
$$N_T = \min\{s \in \{1, 2, \ldots, N\} \,|\, T \in \mathcal{T}_s\}.$$

There are two terms in (3) within the parenthesis. The latter one penalizing the number of terminal nodes $|\pi(T)|$ has been commonly adopted in the existing tree literature. The former one is novel. Intuitively, it penalizes non-sparse models since the number of relevant dimensions $r_T$ appears in the exponent term. In the next section, we will show that this sparsity-inducing term is derived by bounding the VC-dimension of the underlying subgraph of regression functions. Thus it has a very intuitive interpretation.

## 3 Statistical Properties

In this section, we present theoretical properties of the MDRT estimator. Our main technical result is Theorem 1, which provides the nearly optimal rate of the MDRT estimator.

To evaluate the algorithm performance, we use the $L_2$-risk with respect to the Lebesgue measure $\mu(\cdot)$, which is defined as $R(\widehat{f}, f) = \mathbf{E} \sum_{j=1}^{p} \int_{\mathcal{X}} |\widehat{f}_j(\mathbf{x}) - f_j(\mathbf{x})|^2 d\mu(\mathbf{x})$, where $\widehat{f}$ is the function estimate constructed from $n$ observed samples. Note that all the constants appear in this section are generic constants, i.e. their values can change from one line to another in the analysis.

Let $\mathbf{N}_0 = \{0, 1, \ldots\}$ be the set of natural number, we first define the class of $(\alpha, C)$-smooth functions.

**Definition 1** (($\alpha, C$)-smoothness) *Let $\alpha = q + \beta$ for some $q \in \mathbf{N}_0$, $0 < \beta \leq 1$, and let $C > 0$. A function $g : \mathbb{R}^d \to \mathbb{R}$ is called $(\alpha, C)$-smooth if for every $\alpha = (\alpha_1, \ldots, \alpha_d), \alpha_i \in \mathbf{N}_0, \sum_{j=1}^{d} \alpha_j = q$, the partial derivative $\frac{\partial^q g}{\partial x_1^{\alpha_1} \ldots \partial x_d^{\alpha_d}}$ exists and satisfies, for all $\mathbf{x}, \mathbf{z} \in \mathbb{R}^d$,*
$$\left| \frac{\partial^q g(\mathbf{x})}{\partial x_1^{\alpha_1} \ldots \partial x_d^{\alpha_d}} - \frac{\partial^q g(\mathbf{z})}{\partial x_1^{\alpha_1} \ldots \partial x_d^{\alpha_d}} \right| \leq C \cdot \|\mathbf{x} - \mathbf{z}\|_2^\beta.$$

In the following, we denote the class of $(\alpha, C)$-smooth functions by $\mathcal{D}(\alpha, C)$.

**Assumption 1** *We assume $f_1, \ldots, f_p \in \mathcal{D}(\alpha, C)$ for some $\alpha, C > 0$ and for all $j \in \{1, \ldots, p\}$, $f_j(\mathbf{x}) = f_j(\mathbf{x}_S)$ with $r = |S| \ll d$.*

Theorem 3.2 of [9] shows that the lower minimax rate of convergence for class $\mathcal{D}(\alpha, C)$ is exactly the same as that for class of $d$-dimensional Sobolev ball with order $\alpha$ and radius $C$.

**Proposition 1** The proof of this proposition can be found in [9].
$$\liminf_{n\to\infty} \frac{1}{p} \cdot n^{2\alpha/(2\alpha+d)} \inf_{\widehat{f}} \sup_{f_1,\ldots,f_p\in\mathcal{D}(\alpha,C)} R(\widehat{f},f) > 0.$$

Therefore, the lower minimax rate of convergence is $p \cdot n^{-2\alpha/(2\alpha+d)}$. Similarly, if the problem is jointly sparse with the index set $S$ and $r = |S| \ll d$, the best rate of convergence can be improved to $p \cdot n^{-2\alpha/(2\alpha+r)}$ when $S$ is given.

The following is another technical assumption needed for the main theorem.

**Assumption 2** *Let* $1 \leq \gamma < \infty$, *we assume that*
$$\max_{1\leq j\leq p} \sup_{\mathbf{x}} |f_j(\mathbf{x})| \leq \gamma \text{ and } \max_{1\leq i\leq n} \|\mathbf{y}^i\|_\infty \leq \gamma \quad \text{a.s.}$$

This condition is mild. Indeed, we can even allow $\gamma$ to increase with the sample size $n$ at a certain rate. This will not affect the final result. For example, when $\{\epsilon_j^i\}_{i,j}$ are i.i.d. Gaussian random variables, this assumption easily holds with $\gamma = O(\sqrt{\log n})$, which only contributes a logarithmic term to the final rate of convergence.

The next assumption specifies the scaling of the relevant dimension $r$ and ambient dimension $d$ with respect to the sample size $n$.

**Assumption 3** $r = O(1)$ *and* $d = O(\exp(n^\xi))$ *for some* $0 < \xi < 1$.

Here, $r = O(1)$ is crucial, since even if $r$ increases at a logarithmic rate with respect to $n$, i.e. $r = O(\log n)$, it is hopeless to get any consistent estimator for the class $\mathcal{D}(\alpha, C)$ since $n^{-(1/\log n)} = 1/e$. On the other hand, the ambient dimension $d$ can increase exponentially fast with the sample size, which is a realistic scaling for high dimensional settings.

The following is the main theorem.

**Theorem 1** *Under Assumptions 1 to 3, there exist a positive number* $\lambda$ *that only depends on* $\alpha, \gamma$ *and* $r$, *such that*
$$\text{pen}(f) = \lambda \cdot \frac{p}{n}\Big((\log n)(r_T + 1)^m(N_T + 1)^{r_T} + |\pi(T)|\log d\Big), \tag{4}$$
*For large enough* $M, N$, *the solution* $\widehat{f}^{M,N}$ *obtained from* (2) *satisfies*
$$R(\widehat{f}^{M,N}, f) \leq c \cdot p \cdot \left(\frac{\log n + \log d}{n}\right)^{2\alpha/(2\alpha+r)}, \tag{5}$$
*where* $c$ *is some generic constant.*

**Remark 1** *As discussed in Proposition 1, the obtained rate of convergence in* (5) *is nearly optimal up to a logarithmic term.*

**Remark 2** *Since the estimator defined in* (2) *does not need to know the smoothness* $\alpha$ *and the sparsity level* $r$ *in advance, MDRTs are simultaneously adaptive to the unknown smoothness and sparsity level.*

**Proof of Theorem 1**: To find an upper bound of $R(\widehat{f}^{M,N}, f)$, we need to analyze and control the approximation and estimation errors separately. Our analysis closely follows the least squares regression analysis in [9] and some specific coding scheme of trees in [15].

Without loss of generality, we always assume $\widehat{f}^{M,N}$ obtained from (2) satisfies the condition that $\max_{1\leq j\leq p} \sup_{\mathbf{x}} |f_j^{M,N}(\mathbf{x})| \leq \gamma$. if this is not true, we can always truncate $\widehat{f}^{M,N}$ at the rate $\gamma$ and obtain the desired result in Theorem 1.

Let $\mathcal{S}_T^m$ be the class of scalar-valued measurable $m$-th order polynomials corresponding to $\pi(T)$, and let $\mathcal{G}_T^m$ be the class of all subgraphs of functions of $\mathcal{S}_T^m$, i.e.
$$\mathcal{G}_T^m = \big\{(\mathbf{z}, t) \in \mathbb{R}^d \times \mathbb{R}; t \leq g(\mathbf{z}); g \in \mathcal{S}_T^m\big\}.$$
Let $V_{\mathcal{G}_T^m}$ be the VC-dimension of $\mathcal{G}_T^m$, we have the following lemma:

**Lemma 1** *Let $r_T$ and $N_T$ be defined as in* (3), *we know that*
$$V_{\mathcal{G}_T^m} \leq (r_T + 1)^m \cdot (N_T + 1)^{r_T}.$$
(6)

**Sketch of Proof**: From Theorem 9.5 of [9], we only need to show the dimension of $\mathcal{G}_T^m$ is upper bounded by the R.H.S. of (6). By the definition of $r_T$ and $N_T$, the result follows from a straightforward combinatorial analysis. □

The next lemma provides an upper bound of the approximation error for the class $\mathcal{D}(\alpha, C)$.

**Lemma 2** *Let $f = (f_1, \ldots, f_p)$ be the true regression function, there exists a set of piecewise polynomials $h_1, \ldots, h_p \in \cup_{T \in \mathcal{T}_K} S_T^m$*
$$\forall j \in \{1, \ldots, p\}, \; \sup_{\mathbf{x} \in \mathcal{X}} |f_j(\mathbf{x}) - h_j(\mathbf{x})| \leq cK^{-\alpha}$$
*where $K \leq N$, $c$ is a generic constant depends on $r$.*

**Sketch of Proof**: This is a standard approximation result using multivariate piecewise polynomials. The main idea is based on a multivariate Taylor expansion of the function $f_j$ at a given point $\mathbf{x}_0$. Then try to utilize Definition 1 to bound the remainder terms. For the sake of brevity, we omit the technical details. □

The next lemma is crucial, it provides an oracle inequality to bound the risk using an approximation term and an estimation term. Its analysis follows from a simple adaptation of Theorem 12.1 on page 227 of [9].

First, we define $\widetilde{R}(g, f) = \sum_{j=1}^{p} \int_{\mathcal{X}} |g_j(\mathbf{x}) - f_j(\mathbf{x})|^2 d\mu(\mathbf{x})$,

**Lemma 3** [9] *Choose*
$$\text{pen}(f) \geq 5136 \cdot p \frac{\gamma^4}{n} \left( \log(120e\gamma^4 n) V_{\mathcal{G}_T^m} + \frac{[[T]] \log 2}{2} \right)$$
(7)
*for some prefix code $[[T]] > 0$ satisfying $\sum_{T \in \mathcal{T}_N} 2^{-[[T]]} \leq 1$. Then, we have*
$$R(\widehat{f}^{M,N}, f) \leq 12840 \cdot p \cdot \frac{\gamma^4}{n} + 2 \inf_{T \in \mathcal{T}_N} \inf_{g \in \mathcal{F}^{M,N}} \left\{ p \cdot \text{pen}(g) + \widetilde{R}(g, f) \right\}.$$
(8)

One appropriate prefix code $[[T]]$ for each MDRT $T$ is proposed in [15], which specifies that $[[T]] = 3|\pi(T)| - 1 + (|\pi(T)| - 1) \log d / \log 2$. A simpler upper bound for $[[T]]$ is $[[T]] \leq (3 + \log d / \log 2)|\pi(T)|$.

**Remark 3** *The derived constants in the Lemma 3 will be pessimistic due to the very large numerical values. This may result in selecting oversimplified tree structures. In practice, we always use cross-validation to choose the tuning parameters.*

To prove Theorem 1, first, using Assumption 1 and Lemma 2, we know that for any $K \leq N$, there must exists generic constants $c_1, c_2, c_3$ and a function $f'$ that is conformal with a MDRT $T' \in \mathcal{T}_K$, satisfying $f'(\mathbf{x}) = f'(\mathbf{x}_S)$ and $|\pi(T')| \leq (K+1)^r$ such that
$$\widetilde{R}(f', f) \leq c_1 \cdot p \cdot K^{-2\alpha},$$
(9)
and
$$\text{pen}(f') \leq \quad c_2 \frac{(\log n)(r+1)^M (K+1)^r}{n} + c_3 \frac{\log d (K+1)^r}{n}.$$
(10)
The desired result then follows by plugging (9) and (10) into (8) and balancing these three terms.

## 4  Computational Algorithm

Exhaustive search of $\widehat{f}^{M,N}$ in the MDRT space has similar complexity as that of DDTs and could be computationally very expansive. To make MDRTs scalable for high dimensional massive datasets, using similar ideas as CARTs, we propose a two-stage procedure: (1) we grow a full tree in a greedy manner; (2) we prune back the full tree to from the final tree. Before going to the detail of the algorithm, we firstly introduce some necessary notations.

Given a MDRT $T$, denote the corresponding multivariate $m$-th order polynomial fit on $\pi(T)$ by $\widehat{f}_T^m = \{\widehat{f}_t^m\}_{t \in \pi(T)}$, where $\widehat{f}_t^m$ is the $m$-th order polynomial regression fit on the partition $B_t$. For

each $\mathbf{x}^i$ falling in $B_t$, let $\widehat{f}_t^m(\mathbf{x}^i, \mathcal{A}(t))$ be the predicted function value for $\mathbf{x}^i$. We denote the the local squared error (LSE) on node $t$ by $\widehat{R}^m(t, \mathcal{A}(t))$:

$$\widehat{R}^m(t, \mathcal{A}(t)) = \frac{1}{n} \sum_{\mathbf{x}^i \in B_t} \|\mathbf{y}^i - \widehat{f}_t^m(\mathbf{x}^i, \mathcal{A}(t))\|_2^2.$$

It is worthwhile noting that $\widehat{R}^m(t, \mathcal{A}(t))$ is calculated as the average with respect to the total sample size $n$, instead of the number of data points contained in $B_t$. The total MSE of the tree $\widehat{R}(T)$ can then be computed by the following equation:

$$\widehat{R}(T) = \sum_{t \in \text{term}(T)} \widehat{R}^m(t, \mathcal{A}(t)).$$

The total cost of $T$, which is defined as the the right hand side of (2), then can be written as:

$$\widehat{C}(T) = \widehat{R}(T) + \text{pen}(\widehat{f}_T^m). \tag{11}$$

Our goal is to find the tree structure with the polynomial regression on each terminal node that can minimize the total cost.

The first stage is *tree growing*, in which a terminal node $t$ is first selected in each step. We then perform one of two actions a1 and a2:

    a1: adding another dimension $k \notin \mathcal{A}(t)$ to $\mathcal{A}(t)$, and refit the regression model on all data points falling in $B_t$;

    a2: dyadically splitting $t$ perpendicular to the dimension $k \in \mathcal{A}(t)$.

In each tree growing step, we need to decide which action to perform. For action a1, we denote the drop in LSE as:

$$\Delta \widehat{R}_1^m(t, k) = \widehat{R}^m(t, \mathcal{A}(t)) - \widehat{R}^m(t, \mathcal{A}(t) \cup \{k\}). \tag{12}$$

For action a2, let $\text{sl}(t^{(k)})$ be the side length of $B_t$ on dimension $k \in \mathcal{A}(t)$. If $\text{sl}(t^{(k)}) > 2^{-L}$, the dimension $k$ of $B_t$ can then be dyadically split. In this case, let $t_L^{(k)}$ and $t_R^{(k)}$ be the left and right child of node $t$. The drop in LSE takes the following form:

$$\Delta \widehat{R}_2^m(t, k) = \widehat{R}^m(t, \mathcal{A}(t)) - \widehat{R}^m(t_L^{(k)}, \mathcal{A}(t) - \widehat{R}^m(t_R^{(k)}, \mathcal{A}(t)). \tag{13}$$

For each terminal node $t$, we greedily perform the action $a^*$ on the dimension $k^*$, which are determined by

$$(a^*, k^*) = \operatorname*{argmax}_{a \in \{1,2\}, k \in \{1...d\}} \Delta \widehat{R}_a^m(t, k). \tag{14}$$

In high dimensional setting, the above greedy procedure may not lead to the optimal tree since successively locally optimal splits cannot guarantee the global optimum. Once an irrelevant dimension has been added in or split, the greedy procedure can never fix the mistake. To make the algorithm more robust, we propose a randomized scheme. Instead of greedily performing the action on the dimension that leads the maximum drop in LSE, we randomly choose which action to perform according to a multinomial distribution. In particular, we normalize $\Delta \widehat{R}$ such that:

$$\sum_{a=1}^{2} \sum_k \Delta \widehat{R}_a^m(t, k) = 1. \tag{15}$$

And a sample $(a^*, k^*)$ is drawn from multinomial$(1, \Delta \widehat{R})$. The action $a^*$ is then performed on the dimension $k^*$. In general, when the randomized scheme is adopted, we need to repeat our algorithm many times to pick the best tree.

The second stage is *cost complexity pruning*. For each step, we either merge a pair of terminal nodes or remove a variable from the active set of a terminal node such that the resulted tree has the smaller cost. We repeat this process until the tree becomes a single root node with an empty active set. The tree with the minimum cost in this process is returned as the final tree. The pseudocode for the growing stage and cost complexity pruning stage are presented in the Appendix. Moreover, to avoid a cell with too few data points, we pre-define a quantity $n_{\max}$. Let $n(t)$ be the the number of data points fall into $B_t$, if $n(t) \leq n_{\max}$, $B_t$ will no longer be split. It is worthwhile noting that we ignore those actions that lead to $\Delta R = 0$. In addition, whenever we perform the $m$th order polynomial regression on the active set of a node, we need to make sure it is not rank deficient.

# 5 Experimental Results

In this section, we present numerical results for MDRTs applied to both synthetic and real datasets. We compare five methods: [1] Greedy MDRT with $M = 1$ (MDRT(G, M=1)); [2] Randomized MDRT with $M = 1$ (MDRT(R, M=1)); [3] Greedy MDRT with $M = 0$ (MDRT(G, M=0)); [4] Randomized MDRT with $M = 0$ (MDRT(R, M=0)); [5] CART. For randomized scheme, we run 50 random trials and pick the minimum cost tree.

As for CART, we adopt the MATLAB package from [12], which fits piecewise constant on each terminal node with the cost complexity criterion: $\widehat{C}(T) = \widehat{R}(T) + \rho \frac{p}{n} |\pi(T)|$, where $\rho$ is the tuning parameter playing the same role as $\lambda$ in (3).

**Synthetic Data**: For the synthetic data experiment, we consider the high dimensional *compound symmetry* covariance structure of the design matrix with $n = 200$ and $d = 100$. Each dimension $\mathbf{x}_j$ is generated according to
$$\mathbf{x}_j = \frac{W_j + tU}{1 + t}, \quad j = 1, \ldots, d,$$
where $W_1, \ldots, W_d$ and $U$ are i.i.d. sampled from Uniform(0,1). Therefore the correlation between $\mathbf{x}_j$ and $\mathbf{x}_k$ is $t^2/(1 + t^2)$ for $j \neq k$.

We study three models as shown below: the first one is linear; the second one is nonlinear but additive; the third one is nonlinear with three-way interactions. All these models only involve four relevant variables. The noise terms, denoted as $\epsilon$, are independently drawn from a standard normal distribution.

Model 1: $\quad \mathbf{y}_1^i = 2\mathbf{x}_1^i + 3\mathbf{x}_2^i + 4\mathbf{x}_3^i + 5\mathbf{x}_4^i + \epsilon_1^i \qquad \mathbf{y}_2^i = 5\mathbf{x}_1^i + 4\mathbf{x}_2^i + 3\mathbf{x}_3^i + 2\mathbf{x}_4^i + \epsilon_2^i$

Model 2: $\quad \mathbf{y}_1^i = \exp(\mathbf{x}_1^i) + (\mathbf{x}_2^i)^2 + 3\mathbf{x}_3^i + 2\mathbf{x}_4^i + \epsilon_1^i \qquad \mathbf{y}_2^i = (\mathbf{x}_1^i)^2 + 2\mathbf{x}_2^i + \exp(\mathbf{x}_3^i) + 3\mathbf{x}_4^i + \epsilon_2^i$

Model 3: $\quad \mathbf{y}_1^i = \exp(2\mathbf{x}_1^i\mathbf{x}_2^i + \mathbf{x}_3^i) + \mathbf{x}_4^i + \epsilon_1^i \qquad \mathbf{y}_2^i = \sin(\mathbf{x}_1^i\mathbf{x}_2^i) + (\mathbf{x}_3^i)^2 + 2\mathbf{x}_4^i + \epsilon_2^i$

We compare the performances of different methods using two criteria: (i) variable selection and (ii) function estimation. For each model, we generate 100 designs and an equal-sized validation set per design. For more detailed experiment protocols, we set $n_{\max} = 5$ and $L = 6$. By varying the values of $\lambda$ or $\rho$ from large to small, we obtain a full regularization path. The tree with the minimum MSE on the validation set is then picked as the best tree. For criterion (i), if the variables involved in the best tree are exactly the first four variables, the variable selection task for this design is deemed as successful. The numerical results are presented in Table 1. For each method, the three quantities reported in order are the number of success out of 100 designs, the mean and standard deviation of the MSE on the validation set. Note that we omit "MDRT" in Table 1 due to space limitations.

From Table 1, the performance of MDRT with $M = 1$ is dominantly better in both variable selection and estimation than those of the others. For linear models, MDRT with $M = 1$ always select the correct variables even for large $t$s. For variable selection, MDRT with $M = 0$ has a better performance compared with CART due to its sparsity-inducing penalty. In contrast, CART is more flexible in the sense that its splits are not necessarily dyadic. As a consequence, they are comparable in function estimation. Moreover, the performance of randomized scheme is slightly better than its deterministic version in variable selection. Another observation is that, when $t$ becomes larger, although the performance of variable selection decreases on all methods, the estimation performance becomes slightly better. This might be counter-intuitive at the first sight. In fact, with the increase of $t$, all methods tend to select more variables. Due to the high correlations, even the irrelevant variables are also helpful in predicting the responses. This is an expected effect.

**Real Data**: In this subsection, we compare these methods on three real datasets. The first dataset is the *Chemometrics* data (Chem for short), which has been extensively studied in [3]. The data are from a simulation of a low density tubular polyethylene reactor with $n = 56$, $d = 22$ and $p = 6$. Following the same procedures in [3], we log-transformed the responses because they are skewed. The second dataset is Boston *Housing* [1] with $n = 506$, $d = 10$ and $p = 1$. We add 10 irrelevant variables randomly drawn from Uniform(0,1) to evaluate the variable selection performance. The third one, *Space_ga*[2], is an election data with spatial coordinates on 3107 US counties. Our task is to predict the $x, y$ coordinates of each county given 5 variables regarding voting information.

Table 1: Comparison of Variable Selection and Function Estimation on Synthetic Datasets

| Model 1 | R, M=1 | | G, M=1 | | R, M=0 | | G, M=0 | | CART | |
|---|---|---|---|---|---|---|---|---|---|---|
| $t = 0$ | 100 | 2.03 (0.14) | 100 | 2.08 (0.15) | 100 | 5.84 (0.51) | 97 | 5.74 (0.54) | 52 | 6.17 (0.55) |
| $t = 0.5$ | 100 | 2.05 (0.14) | 100 | 2.06 (0.15) | 76 | 5.42 (0.53) | 68 | 5.36 (0.60) | 29 | 5.48 (0.51) |
| $t = 1$ | 100 | 2.05 (0.13) | 100 | 2.05 (0.16) | 19 | 5.40 (0.60) | 20 | 5.56 (0.69) | 3 | 5.30 (0.58) |

| Model 2 | R, M=1 | | G, M=1 | | R, M=0 | | G, M=0 | | CART | |
|---|---|---|---|---|---|---|---|---|---|---|
| $t = 0$ | 100 | 2.07 (0.13) | 100 | 2.06 (0.15) | 39 | 3.21 (0.26) | 31 | 3.22 (0.28) | 25 | 3.52 (0.31) |
| $t = 0.5$ | 96 | 2.05 (0.15) | 93 | 2.09 (0.17) | 17 | 3.10 (0.25) | 11 | 3.15 (0.26) | 5 | 3.20 (0.27) |
| $t = 1$ | 76 | 2.09 (0.14) | 68 | 2.21 (0.19) | 2 | 3.17 (0.30) | 2 | 3.16 (0.26) | 1 | 3.16 (0.27) |

| Model 3 | R, M=1 | | G, M=1 | | R, M=0 | | G, M=0 | | CART | |
|---|---|---|---|---|---|---|---|---|---|---|
| $t = 0$ | 98 | 2.68 (0.31) | 95 | 2.67 (0.47) | 75 | 3.90 (0.47) | 63 | 4.03 (0.54) | 29 | 4.35 (0.73) |
| $t = 0.5$ | 84 | 2.56 (0.21) | 86 | 2.52 (0.25) | 32 | 3.63 (0.47) | 32 | 3.60 (0.40) | 15 | 3.69 (0.38) |
| $t = 1$ | 65 | 2.51 (0.26) | 50 | 2.62 (0.23) | 3 | 3.75 (0.45) | 4 | 3.88 (0.51) | 2 | 3.66 (0.38) |

For Space_ga, we normalize the responses to $[0, 1]$. Similarly, we add other 15 irrelevant variables randomly drawn from Uniform(0,1). For all these datasets, we scale the input variables into a unit cube.

For evaluation purpose, each dataset is randomly split such that half data are used for training and the other half for testing. We run a 5-fold cross-validation on the training set to pick the best tuning parameter $\lambda^*$ and $\rho^*$. We then train MDRTs and CART on the entire training data using $\lambda^*$ and $\rho^*$. We repeat this process 20 times and report the mean and standard deviation of the testing MSE in Table 2. $n_{\max}$ is set to be 5 for the first dataset and 20 for the latter two. For all datasets, we set $L = 6$. Moreover, for randomized scheme, we run 50 random trials and pick the minimum cost tree.

Table 2: Testing MSE on Real Datasets

| | R, M=1 | G, M=1 | R, M=0 | G, M=0 | CART |
|---|---|---|---|---|---|
| Chem | 0.15 (0.09) | 0.18 (0.12) | 0.38 (0.18) | 0.52 (0.06) | 0.40 (0.09) |
| Housing | 20.18 (2.94) | 21.60 (2.83) | 24.67 (2.05) | 29.46 (1.95) | 25.91 (3.05) |
| Space_ga | 0.054 (7.8e-4) | 0.055 (8.0e-4) | 0.068 (7.2e-4) | 0.068 (9.2e-4) | 0.064 (8.3e-4) |

From Table 2, we see that MDRT with $M = 1$ has the best estimation performance. Moreover, randomized scheme does improve the performance compared to the deterministic counterpart. In particularly, such an improvement is quite significant when $M = 0$. The performance of MDRT(G, M=0) is always worse than CART since CART can have more flexible splits. However, using randomized scheme, the performance of MDRT(R, M=0) achieves a comparable performance as CART.

As for variable selection of Housing data, in all the 20 runs, MDRT(G, M=1) and MDRT(R, M=1) never select the artificially added variables. However, for the other three methods, nearly 10 out of 20 runs involve at least one extraneous variable. In particular, we compare our results with those reported in [14]. They find that there are 4 (indus, age, dis, tax) irrelevant variables in the Housing data. Our experiments confirm this result since in 15 out of the 20 trials, MDRT(G, M=1) and MDRT(R, M=1) never select these four variables. Similarly, for Space_ga data, there are only 2 and 1 times that MDRT(G, M=1) and MDRT(R, M=1) involve the artificially added variables.

## 6 Conclusions

We propose a novel sparse learning method based on multivariate dyadic regression trees (MDRTs). Our approach adopts a new sparsity-inducing penalty that simultaneously conduct function estimation and variable selection. Some theoretical analysis and practical algorithms have been developed. To the best of our knowledge, it is the first time that such a penalty is introduced in the tree literature for high dimensional sparse learning problems.

## Footnotes

[1] Available from *UCI Machine Learning Database Repository*: `http:archive.ics.uci.edu/ml`

[2] Available from *StatLib*: `http:lib.stat.cmu.edu/datasets/`

# References

[1] G. Blanchard, C. Schäfer, Y. Rozenholc, and K.-R. Müller. Optimal dyadic decision trees. *Machine Learning Journal*, 66(2-3):209–241, 2007.

[2] Leo Breiman, Jerome Friedman, Charles J. Stone, and R.A. Olshen. *Classification and regression trees*. Wadsworth Publishing Co Inc, 1984.

[3] Leo Breiman and Jerome H. Friedman. Predicting multivariate responses in multiple linear regression. *J. Roy. Statist. Soc. B*, 59:3, 1997.

[4] R. Castro, R. Willett, and R. Nowak. Fast rates in regression via active learning. *NIPS*, 2005.

[5] Xi Chen, Weike Pan, James T. Kwok, and Jamie G. Carbonell. Accelerated gradient method for multi-task sparse learning problem. In *ICDM*, 2009.

[6] Hugh A. Chipman, Edward I. George, and Robert E. McCulloch. Bart: Bayesian additive regression trees. Technical report, Department of Mathematics and Statistics, Acadia University, Canada, 2006.

[7] Jerome H. Friedman. Multivariate adaptive regression splines. *The Annals of Statistics*, 19:1–67, 1991.

[8] S. Gey and E. Nedelec. Model selection for cart regression trees. *IEEE Tran. on Info. Theory*, 51(2):658– 670, 2005.

[9] László Györfi, Michael Kohler, Adam Krzyżak, and Harro Walk. *A Distribution-Free Theory of Nonparametric Regression*. Springer-Verlag, 2002.

[10] Han Liu, John Lafferty, and Larry Wasserman. Nonparametric regression and classification with joint sparsity constraints. In *NIPS*. MIT Press, 2008.

[11] Han Liu and Jian Zhang. On the estimation consistency of the group lasso and its applications. *AISTATS*, pages 376–383, 2009.

[12] Wendy L. Martinez and Angel R. Martinez. *Computational Statistics Handbook with MATLAB*. Chapman & Hall CRC, 2 edition, 2008.

[13] G. Obozinski, M. J. Wainwright, and M. I. Jordan. High-dimensional union support recovery in multivariate regression. In *NIPS*. MIT Press, 2009.

[14] Pradeep Ravikumar, Han Liu, John Lafferty, and Larry Wasserman. Spam: Sparse additive models. In *NIPS*. MIT Press, 2007.

[15] C. Scott and R.D. Nowak. Minimax-optimal classification with dyadic decision trees. *IEEE Tran. on Info. Theory*, 52(4):1335–1353, 2006.

[16] B.A. Turlach, W. N. Venables, and S. J. Wright. Simultaneous variable selection. *Technometrics*, 27:349–363, 2005.

